# PREDICTIVE CODING WITH NEURAL NETS: APPLICATION TO TEXT COMPRESSION

Jürgen Schmidhuber                    Stefan Heil

Fakultät für Informatik
Technische Universität München
80290 München, Germany

## Abstract

To compress text files, a neural predictor network $P$ is used to approximate the conditional probability distribution of possible "next characters", given $n$ previous characters. $P$'s outputs are fed into standard coding algorithms that generate short codes for characters with high predicted probability and long codes for highly unpredictable characters. Tested on short German newspaper articles, our method outperforms widely used Lempel-Ziv algorithms (used in UNIX functions such as "compress" and "gzip").

# 1    INTRODUCTION

The method presented in this paper is an instance of a strategy known as "predictive coding" or "model-based coding". To compress text files, a neural predictor network $P$ approximates the conditional probability distribution of possible "next characters", given $n$ previous characters. $P$'s outputs are fed into algorithms that generate short codes for characters with low information content (characters with high predicted probability) and long codes for characters conveying a lot of information (highly unpredictable characters) [5]. Two such standard coding algorithms are employed: Huffman Coding (see e.g. [1]) and Arithmetic Coding (see e.g. [7]).

With the *off-line* variant of the approach, $P$'s training phase is based on a set $F$ of training files. After training, the weights are frozen. Copies of $P$ are installed at all machines functioning as message receivers or senders. From then on, $P$ is used to encode and decode unknown files without being changed any more. The weights become part of the code of the compression algorithm. Note that the storage occupied by the network weights does not have to be taken into account to measure the performance on *unknown* files – just like the code for a conventional data compression algorithm does not have to be taken into account.

The more sophisticated *on-line* variant of our approach will be addressed later.

# 2    A PREDICTOR OF CONDITIONAL PROBABILITIES

Assume that the alphabet contains $k$ possible characters $z_1, z_2, \ldots, z_k$. The (local) representation of $z_i$ is a binary $k$-dimensional vector $r(z_i)$ with exactly one non-zero component (at the $i$-th position). $P$ has $nk$ input units and $k$ output units. $n$ is called the "time-window size". We insert $n$ default characters $z_0$ at the beginning of each file. The representation of the default character, $r(z_0)$, is the $k$-dimensional zero-vector. The $m$-th character of file $f$ (starting from the first default character) is called $c_m^f$.

For all $f \in F$ and all possible $m > n$, $P$ receives as an input

$$r(c_{m-n}^f) \circ r(c_{m-n+1}^f) \circ \ldots \circ r(c_{m-1}^f), \tag{1}$$

where $\circ$ is the concatenation operator for vectors. $P$ produces as an output $P_m^f$, a $k$-dimensional output vector. Using back-propagation [6][2][3][4], $P$ is trained to minimize

$$\frac{1}{2} \sum_{f \in F} \sum_{m > n} \| r(c_m^f) - P_m^f \|^2 . \tag{2}$$

Expression (2) is minimal if $P_m^f$ always equals

$$E(r(c_m^f) \mid c_{m-n}^f, \ldots, c_{m-1}^f), \tag{3}$$

the conditional expectation of $r(c_m^f)$, given $r(c_{m-n}^f) \circ r(c_{m-n+1}^f) \circ \ldots \circ r(c_{m-1}^f)$. Due to the local character representation, this is equivalent to $(P_m^f)_i$ being equal to the

conditional probability

$$Pr(c_m^f = z_i \mid c_{m-n}^f, \ldots, c_{m-1}^f) \qquad (4)$$

for all $f$ and for all appropriate $m > n$, where $(P_m^f)_j$ denotes the $j$-th component of the vector $P_m^f$.

In general, the $(P_m^f)_i$ will not quite match the corresponding conditional probabilities. For normalization purposes, we define

$$P_m^f(i) = \frac{(P_m^f)_i}{\sum_{j=1}^{k}(P_m^f)_j}. \qquad (5)$$

No normalization is used during training, however.

# 3   HOW TO USE THE PREDICTOR FOR COMPRESSION

We use a standard procedure for predictive coding. With the help of a copy of $P$, an unknown file $f$ can be compressed as follows: Again, $n$ default characters are inserted at the beginning. For each character $c_m^f$ ($m > n$), the predictor emits its output $P_m^f$ based on the $n$ previous characters. There will be a $k$ such that $c_m^f = z_k$. The estimate of $P(c_m^f = z_k \mid c_{m-n}^f, \ldots, c_{m-1}^f)$ is given by $P_m^f(k)$. The code of $c_m^f$, $code(c_m^f)$, is generated by feeding $P_m^f(k)$ into the Huffman Coding algorithm (see below), or, alternatively, into the Arithmetic Coding algorithm (see below). $code(c_m^f)$ is written into the compressed file. The basic ideas of both coding algorithms are described next.

## 3.1   HUFFMAN CODING

With a given probability distribution on a set of possible characters, Huffman Coding (e.g. [1]) encodes characters by bitstrings as follows.

Characters are terminal nodes of a binary tree to be built in an incremental fashion. The probability of a terminal node is defined as the probability of the corresponding character. The probability of a non-terminal node is defined as the sum of the probabilities of its sons. Starting from the terminal nodes, a binary tree is built as follows:

> **Repeat as long as possible:**
> *Among those nodes that are not children of any non-terminal nodes created earlier, pick two with lowest associated probabilities. Make them the two sons of a newly generated non-terminal node.*

The branch to the "left" son of each non-terminal node is labeled by a 0. The branch to its "right" son is labeled by a 1. The code of a character $c$, $code(c)$, is the bitstring obtained by following the path from the root to the corresponding node. Obviously, if $c \neq d$, then $code(c)$ cannot be the prefix of $code(d)$. This makes the code uniquely decipherable.

Characters with high associated probability are encoded by short bitstrings. Characters with low associated probability are encoded by long bitstrings. Huffman Coding guarantees minimal expected code length, provided all character probabilities are integer powers of $\frac{1}{2}$.

## 3.2   ARITHMETIC CODING

In general, Arithmetic Coding works slightly better than Huffman Coding. For sufficiently long messages, Arithmetic Coding achieves expected code lenghts arbitrarily close to the information-theoretic lower bound. This is true even if the character probabilities are not powers of $\frac{1}{2}$ (see e.g. [7]).

The basic idea of Arithmetic Coding is: a message is encoded by an interval of real numbers from the unit interval $[0, 1[$. The output of Arithmetic Coding is a binary representation of the boundaries of the corresponding interval. This binary representation is incrementally generated during message processing. Starting with the unit interval, for each observed character the interval is made smaller, essentially in proportion to the probability of the character. A message with low information content (and high corresponding probability) is encoded by a comparatively large interval whose precise boundaries can be specified with comparatively few bits. A message with a lot of information content (and low corresponding probability) is encoded by a comparatively small interval whose boundaries require comparatively many bits to be specified.

Although the basic idea is elegant and simple, additional technical considerations are necessary to make Arithmetic Coding practicable. See [7] for details.

Neither Huffman Coding nor Arithmetic Coding requires that the probability distribution on the characters remains fixed. This allows for using "time-varying" conditional probability distributions as generated by the neural predictor.

## 3.3   HOW TO "UNCOMPRESS" DATA

The information in the compressed file is sufficient to reconstruct the original file without loss of information. This is done with the "uncompress" algorithm, which works as follows: Again, for each character $c_m^f$ ($m > n$), the predictor (sequentially) emits its output $P_m^f$ based on the $n$ previous characters, where the $c_l^f$ with $n < l < m$ were gained sequentially by feeding the approximations $P_l^f(k)$ of the probabilities $P(c_l^f = z_k \mid c_{l-n}^f, \ldots, c_{l-1}^f)$ into the inverse Huffman Coding procedure (see e.g. [1]), or, alternatively (depending on which coding procedure was used), into the inverse Arithmetic Coding procedure ( e.g. [7]). Both variants allow for correct decoding of $c_l^f$ from $code(c_l^f)$. With both variants, to correctly decode some character, we first need to decode all previous characters. Both variants are **guaranteed** to restore the original file from the compressed file.

## WHY NOT USE A LOOK-UP TABLE INSTEAD OF A NETWORK?

Because a look-up table would be extremely inefficient. A look-up table requires $k^{n+1}$ entries for all the conditional probabilities corresponding to all possible com-

binations of $n$ previous characters and possible next characters. In addition, a special procedure is required for dealing with previously unseen combinations of input characters. In contrast, the size of a neural net typically grows in proportion to $n^2$ (assuming the number of hidden units grows in proportion to the number of input units), and its inherent "generalization capability" is going to take care of previously unseen combinations of input characters (hopefully by coming up with good predicted probabilities).

# 4 SIMULATIONS

We implemented both alternative variants of the encoding and decoding procedure described above.

Our current computing environment prohibits extensive experimental evaluations of the method. The predictor updates turn out to be quite time consuming, which makes special neural net hardware recommendable. The limited software simulations presented in this section, however, will show that the "neural" compression technique can achieve "excellent" compression ratios. Here the term "excellent" is defined by a statement from [1]:

> "In general, good algorithms can be expected to achieve an average compression ratio of 1.5, while excellent algorithms based upon sophisticated processing techniques will achieve an average compression ratio exceeding 2.0."

Here the average compression ratio is the average ratio between the lengths of original and compressed files.

The method was applied to German newspaper articles. The results were compared to those obtained with standard encoding techniques provided by the operating system UNIX, namely "pack", "compress", and "gzip". The corresponding decoding algorithms are "unpack", "uncompress", and "gunzip", respectively. "pack" is based on Huffman-Coding (e.g. [1]), while "compress" and "gzip" are based on techniques developed by Lempel and Ziv (e.g. [9]). As the file size goes to infinity, Lempel-Ziv becomes *asymptotically optimal* in a certain information theoretic sense [8]. This does not necessarily mean, however, that Lempel-Ziv is optimal for finite file sizes.

The training set for the predictor was given by a set of 40 articles from the newspaper *Münchner Merkur*, each containing between 10000 and 20000 characters. The alphabet consisted of $k = 80$ possible characters, including upper case and lower case letters, digits, interpunction symbols, and special German letters like "ö", "ü", "ä". $P$ had 430 hidden units. A "true" unit with constant activation 1.0 was connected to all hidden and output units. The learning rate was 0.2. The training phase consisted of 25 sweeps through the training set.

The test set consisted of newspaper articles excluded from the training set, each containing between 10000 and 20000 characters. Table 1 lists the average compression ratios. The "neural" method outperformed the strongest conventional competitor, the UNIX "gzip" function based on a Lempel-Ziv algorithm.

| Method | Compression Ratio |
|---|---|
| Huffman Coding (UNIX: pack) | 1.74 |
| Lempel-Ziv Coding (UNIX: compress) | 1.99 |
| Improved Lempel-Ziv ( UNIX: gzip -9) | 2.29 |
| Neural predictor + Huffman Coding, $n = 5$ | 2.70 |
| Neural predictor + Arithmetic Coding, $n = 5$ | 2.72 |

Table 1: *Compression ratios of various compression algorithms for short German text files ($< 20000$ Bytes) from the unknown test set.*

| Method | Compression Ratio |
|---|---|
| Huffman Coding (UNIX: pack) | 1.67 |
| Lempel-Ziv Coding (UNIX: compress) | 1.71 |
| Improved Lempel-Ziv ( UNIX: gzip -9) | 2.03 |
| Neural predictor + Huffman Coding, $n = 5$ | 2.25 |
| Neural predictor + Arithmetic Coding, $n = 5$ | 2.20 |

Table 2: *Compression ratios for articles from a different newspaper. The neural predictor was not retrained.*

How does a neural net trained on articles from *Münchner Merkur* perform on articles from other sources? *Without retraining the neural predictor*, we applied all competing methods to 10 articles from another German newspaper (the *Frankenpost*). The results are given in table 2.

The *Frankenpost* articles were harder to compress for all algorithms. But relative performance remained comparable.

Note that the time-window was quite small ($n = 5$). In general, larger time windows will make more information available to the predictor. In turn, this will improve the prediction quality and increase the compression ratio. Therefore we expect to obtain even better results for $n > 5$ and for recurrent predictor networks.

## 5   DISCUSSION / OUTLOOK

Our results show that neural networks are promising tools for loss-free data compression. It was demonstrated that even *off-line* methods based on small time windows can lead to excellent compression ratios – at least with small text files, they can outperform conventional standard algorithms. We have hardly begun, however, to exhaust the potential of the basic approach.

### 5.1   ON-LINE METHODS

A disadvantage of the off-line technique above is that it is off-line: The predictor does not adapt to the specific text file it sees. This limitation is not essential, however. It is straight-forward to construct an *on-line* variant of the approach.

With the on-line variant, the predictor continues to learn *during* compression. The on-line variant proceeds like this: Both the sender and the receiver start with *exactly the same* initial predictor. Whenever the sender sees a new character, it encodes it using its current predictor. The code is sent to the receiver who decodes it. Both the sender and the receiver use *exactly the same* learning protocol to modify their weights. *This implies that the modified weights need **not** be sent from the sender to the receiver and do not have to be taken into account to compute the average compression ratio.* Of course, the on-line method promises much higher compression ratios than the off-line method.

## 5.2 LIMITATIONS

The main disadvantage of both on-line and off-line variants is their computational complexity. The current off-line implementation is clearly slower than conventional standard techniques, by about three orders of magnitude (but no attempt was made to optimize the code with respect to speed). And the complexity of the on-line method is even worse (the exact slow-down factor depends on the precise nature of the learning protocol, of course). For this reason, especially the promising on-line variants can be recommended only if special neural net hardware is available. Note, however, that there are *many* commercial data compression applications which rely on specialized electronic chips.

## 5.3 ONGOING RESEARCH

There are a few obvious directions for *ongoing experimental research:* (1) Use larger time windows – they seem to be promising even for off-line methods (see the last paragraph of section 4). (2) Thoroughly test the potential of on-line methods. Both (1) and (2) should greatly benefit from fast hardware. (3) Compare performance of predictive coding based on neural predictors to the performance of predictive coding based on different kinds of predictors.

## 6 ACKNOWLEDGEMENTS

Thanks to David MacKay for directing our attention towards Arithmetic Coding. Thanks to Margit Kinder, Martin Eldracher, and Gerhard Weiss for useful comments.

# References

[1] G. Held. *Data Compression*. Wiley and Sons LTD, New York, 1991.

[2] Y. LeCun. Une procédure d'apprentissage pour réseau à seuil asymétrique. *Proceedings of Cognitiva 85, Paris*, pages 599–604, 1985.

[3] D. B. Parker. Learning-logic. Technical Report TR-47, Center for Comp. Research in Economics and Management Sci., MIT, 1985.

[4] D. E. Rumelhart, G. E. Hinton, and R. J. Williams. Learning internal representations by error propagation. In *Parallel Distributed Processing*, volume 1, pages 318–362. MIT Press, 1986.

[5] J. H. Schmidhuber and S. Heil. Sequential neural text compression. *IEEE Transactions on Neural Networks*, 1994. Accepted for publication.

[6] P. J. Werbos. *Beyond Regression: New Tools for Prediction and Analysis in the Behavioral Sciences*. PhD thesis, Harvard University, 1974.

[7] I. H. Witten, R. M. Neal, and J. G. Cleary. Arithmetic coding for data compression. *Communications of the ACM*, 30(6):520–540, 1987.

[8] A. Wyner and J. Ziv. Fixed data base version of the Lempel-Ziv data compression algorithm. *IEEE Transactions Information Theory*, 37:878–880, 1991.

[9] J. Ziv and A. Lempel. A universal algorithm for sequential data compression. *IEEE Transactions on Information Theory*, IT-23(5):337–343, 1977.